# Removing Noise in On-Line Search using Adaptive Batch Sizes

**Genevieve B. Orr**
Department of Computer Science
Willamette University
900 State Street
Salem, Oregon 97301
*gorr@willamette.edu*

## Abstract

Stochastic (on-line) learning can be faster than batch learning. However, at late times, the learning rate must be annealed to remove the noise present in the stochastic weight updates. In this annealing phase, the convergence rate (in mean square) is at best proportional to $1/\tau$ where $\tau$ is the number of input presentations. An alternative is to increase the batch size to remove the noise. In this paper we explore convergence for LMS using 1) small but fixed batch sizes and 2) an adaptive batch size. We show that the best adaptive batch schedule is exponential and has a rate of convergence which is the same as for annealing, i.e., at best proportional to $1/\tau$.

## 1 Introduction

Stochastic (on-line) learning can speed learning over its batch training particularly when data sets are large and contain redundant information [Møl93]. However, at late times in learning, noise present in the weight updates prevents complete convergence from taking place. To reduce the noise, the learning rate is slowly decreased (annealed) at late times. The optimal annealing schedule is asymptotically proportional to $\frac{1}{t}$ where $t$ is the iteration [Gol87, LO93, Orr95]. This results in a rate of convergence (in mean square) that is also proportional to $\frac{1}{t}$.

An alternative method of reducing the noise is to simply switch to (noiseless) batch mode when the noise regime is reached. However, since batch mode can be slow, a better idea is to slowly increase the batch size starting with 1 (pure stochastic) and slowly increasing it only "as needed" until it reaches the training set size (pure batch). In this paper we 1) investigate the convergence behavior of LMS when

using small fixed batch sizes, 2) determine the best schedule when using an adaptive batch size at each iteration, 3) analyze the convergence behavior of the adaptive batch algorithm, and 4) compare this convergence rate to the alternative method of annealing the learning rate.

Other authors have approached the problem of redundant data by also proposing techniques for training on subsets of the data. For example, Pluto [PW93] uses active data selection to choose a concise subset for training. This subset is slowly added to over time as needed. Moller [Møl93] proposes combining scaled conjugate gradient descent (SCG) with what he refers to as blocksize updating. His algorithm uses an iterative approach and assumes that the block size does not vary rapidly during training. In this paper, we take the simpler approach of just choosing exemplars *at random* at each iteration. Given this, we then analyze in detail the convergence behavior. Our results are more of theoretical than practical interest since the equations we derive are complex functions of quantities such as the Hessian that are impractical to compute.

## 2   Fixed Batch Size

In this section we examine the convergence behavior for LMS using a fixed batch size. We assume that we are given a large but finite sized training set $T \equiv \{z_i \equiv (x_i, d_i)\}_{i=1}^N$ where $x_i \in \mathcal{R}^m$ is the $i^{th}$ input and $d_i \in \mathcal{R}$ is the corresponding target. We further assume that the targets are generated using a signal plus noise model so that we can write

$$d_i = w_*^T x_i + \epsilon_i \tag{1}$$

where $\omega_* \in \mathcal{R}^m$ is the optimal weight vector and the $\epsilon_i$ is zero mean noise. Since the training set is assumed to be large we take the average of $\epsilon_i$ and $x_i \epsilon_i$ over the training set to be approximately zero. Note that we consider only the problem of optimization of the $\omega$ *over the training set* and do not address the issue of obtaining good generalization over the distribution from which the training set was drawn.

At each iteration, we assume that exactly $n$ samples are randomly drawn *without replacement* from $T$ where $1 \leq n \leq N$. We denote this batch of size $n$ drawn at time $t$ by $B_n(t) \equiv \{z_{k_i}\}_{i=1}^n$. When $n = 1$ we have pure on-line training and when $n = N$ we have pure batch. We choose to sample without replacement so that as the batch size is increased, we have a smooth transition from on-line to batch.

For LMS, the squared error at iteration $t$ *for a batch* of size $n$ is

$$\mathcal{E}_{B_n(t)} \equiv \frac{1}{n} \sum_{z_i \in B_n(t)} \mathcal{E}(z_i) \quad \text{where} \quad \mathcal{E}(z_i) = \frac{1}{2}(d_i - \omega_t^T x_i)^2 \tag{2}$$

and where $\omega_t \in \mathcal{R}^m$ is the current weight in the network. The update weight equation is then $\omega_{t+1} = \omega_t - \mu \frac{\partial \mathcal{E}_{B_n}}{\partial \omega_t}$ where $\mu$ is the fixed learning rate. Rewriting this in terms of the weight error $v \equiv \omega - \omega_*$ and defining $g_{B_n,t} \equiv \partial \mathcal{E}_{B_n(t)}/\partial v_t$ we obtain

$$v_{t+1} = v_t + \frac{\mu}{n} \sum_{z_i \in B_n} (\epsilon_i - v_t^T x_i)x_i = v_t - \mu\, g_{B_n,t}. \tag{3}$$

Convergence (in mean square) to $\omega_*$ can be characterized by the rate of change of the average squared norm of the weight error $E[v^2]$ where $v^2 \equiv v^T v$. From (3) we obtain an expression for $v_{t+1}^2$ in terms of $v_t$,

$$v_{t+1}^2 = v_t^2 - 2\mu\, v_t^T g_{B_n,t} + \mu^2\, g_{B_n,t}^2. \tag{4}$$

To compute the expected value of $v_{t+1}^2$ conditioned on $v_t$ we can average the right side of (4) over all possible ways that the batch $B_n(t)$ can be chosen from the $N$ training examples. In appendix A, we show that

$$\langle g_{B_n,t} \rangle_B = \langle g_{i,t} \rangle_N \tag{5}$$

$$\langle g_{B_n,t}^2 \rangle_B = \frac{N-n}{n(N-1)} \langle g_{i,t}^2 \rangle_N + \frac{(n-1)N}{(N-1)n} \langle g_{i,t} \rangle_N^2 \tag{6}$$

where $\langle \cdot \rangle_N$ denotes average over all examples *in the training set*, $\langle \cdot \rangle_B$ denotes average *over the possible batches* drawn at time $t$, and $g_{i,t} \equiv \partial \mathcal{E}(z_i)/\partial v_t$. The averages over the entire training set are

$$\langle g_{i,t} \rangle_N = \frac{1}{N} \sum_{i=1}^{N} \frac{\partial \mathcal{E}(z_i)}{\partial v_t} = -\sum_{i=1}^{N} \epsilon_i x_i - v_t^T x_i x_i = R v_t \tag{7}$$

$$\langle g_{i,t}^2 \rangle_N = \frac{1}{N} \sum_{i=1}^{N} (\epsilon_i x_i - v_t^T x_i x_i)^T (\epsilon_i x_i - v_t^T x_i x_i) = \sigma_\epsilon^2 (\text{Tr } R) + v_t^T S v_t \tag{8}$$

where $R \equiv \langle xx^T \rangle_N$, $S \equiv \langle xx^T xx^T \rangle_N$[1], $\sigma_\epsilon^2 \equiv \langle \epsilon^2 \rangle$ and $(\text{Tr } R)$ is the trace of $R$. These equations together with (5) and (6) in (4) gives the expected value of $v_{t+1}$ conditioned on $v_t$

$$\langle v_{t+1}^2 | v_t \rangle = v_t^T \left\{ I - 2\mu R + \mu^2 \left( \frac{N(n-1)}{(N-1)n} R^2 + \frac{N-n}{(N-1)n} S \right) \right\} v_t + \frac{\mu^2 \sigma_\epsilon^2 (\text{Tr } R)(N-n)}{n(N-1)}. \tag{9}$$

Note that this reduces to the standard stochastic and batch update equations when $n = 1$ and $n = N$, respectively.

### 2.0.1   Special Cases: 1-D solution and Spherically Symmetric

In 1-dimension we can average over $v_t$ in (9) to give

$$\langle v_{t+1}^2 \rangle = \alpha \langle v_t^2 \rangle + \beta \tag{10}$$

where

$$\alpha = 1 - 2\mu R + \mu^2 \left( \frac{N(n-1)}{(N-1)n} R^2 + \frac{N-n}{(N-1)n} S \right), \quad \beta = \frac{\mu^2 \sigma_\epsilon^2 R (N-n)}{n(N-1)} \tag{11}$$

and where $R$ and $S$ simplify to $R = \langle x^2 \rangle_N$, $S = \langle x^4 \rangle_N$. This is a difference equation which can be solved exactly to give

$$\langle v_t^2 \rangle = \alpha^{t-t_0} \langle v_0^2 \rangle + \frac{1 - \alpha^{t-t_0}}{1 - \alpha} \beta \tag{12}$$

where $\langle v_0^2 \rangle$ is the expected squared weight error at the initial time $t_0$.

Figure 1a compares equation (12) with simulations of 1-D LMS with gaussian inputs for $N = 1000$ and batch sizes $n = 10, 100,$ and $500$. As can be seen, the agreement is good. Note that $\langle v^2 \rangle$ decreases exponentially until flattening out. The equilibrium value can be computed from (12) by setting $t = \infty$ (assuming $|\alpha| < 1$) to give

$$\langle v^2 \rangle_\infty = \frac{\beta}{1 - \alpha} = \frac{\mu \sigma_\epsilon^2 R (N-n)}{2Rn(N-1) - \mu(N(n-1)R^2 + (N-n)S)}. \tag{13}$$

Note that $\langle v^2 \rangle_\infty$ decreases as $n$ increases and is zero only if $n = N$.

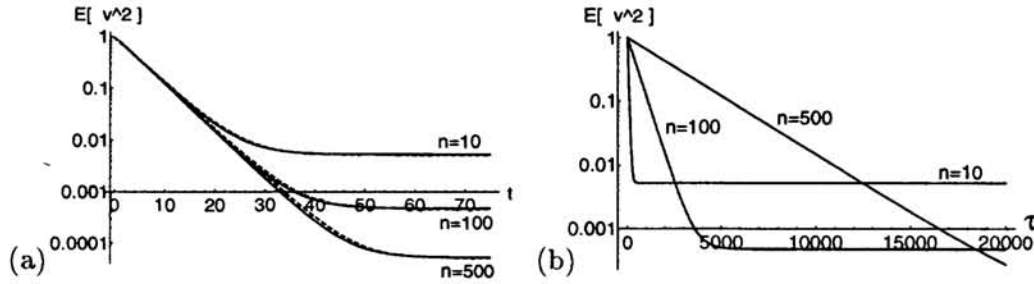

Figure 1: Simulations(solid) vs Theoretical (dashed) predictions of the squared weight error of 1-D LMS a) as function of the number of $t$, the batch updates (iterations), and b) as function of the number of input presentations, $\tau$. Training set size is $N = 1000$ and batch sizes are $n = 10$, 100, and 500. Inputs were gaussian with $R = 1$, $\sigma_\epsilon = 1$ and $\mu = .1$. Simulations used 10000 networks

Equation (9) can also be solved exactly in multiple dimensions in the rather restrictive case where we assume that the inputs are spherically symmetric gaussians with $R = aI$ where $a$ is a constant, $I$ is the identity matrix, and $m$ is the dimension. The update equation and solution are the same as (10) and (12), respectively, but where $\alpha$ and $\beta$ are now

$$\alpha = 1 - 2\mu a + \mu^2 a^2 \left( \frac{N(n-1)}{(N-1)n} + \frac{N-n}{(N-1)n}(m+2) \right), \beta = \frac{\mu^2 \sigma_\epsilon^2 ma(N-n)}{n(N-1)}. \quad (14)$$

## 3  Adaptive Batch Size

The time it takes to compute the weight update in one iteration is roughly proportional to the number of input presentations, i.e the batch size. To make the comparison of convergence rates for different batch sizes meaningful, we must compute the change in squared weight error as a function of the number of input presentations, $\tau$, rather than iteration number $t$.

For fixed batch size, $\tau = nt$. Figure 1b displays our 1-D LMS simulations plotted as a function of $\tau$. As can be seen, training with a large batch size is slow but results in a lower equilibrium value than obtained with a small batch size. This suggests that we could obtain the fastest decrease of $\langle v^2 \rangle$ overall by varying the batch size at each iteration. The batch size to choose for the current $\langle v^2 \rangle$ would be the *smallest* $n$ that has yet to reach equilibrium, i.e. for which $\langle v^2 \rangle > \langle v^2 \rangle_\infty$.

To determine the best batch size, we take the greedy approach by demanding that *at each iteration* the batch size is chosen so as to reduce the weight error at the next iteration by the greatest amount per input presentation. This is equivalent to asking what value of $n$ maximizes $h \equiv (\langle v_t^2 \rangle - \langle v_{t+1}^2 \rangle)/n$? Once we determine $n$ we then express it as a function of $\tau$.

We treat the 1-D case, although the analysis would be similar for the spherically symmetric case. From (10) we have $h = \frac{1}{n} \left( (\alpha - 1)\langle v_t^2 \rangle + \beta \right)$. Differentiating $h$ with respect to $n$ and solving yields the batch size that decreases the weight error the most to be

$$n_t = \min \left( N, \frac{2\mu N((S - R^2)\langle v_t^2 \rangle + \sigma_\epsilon^2 R)}{(2R(N-1) + \mu(S - NR^2))\langle v_t^2 \rangle + \mu \sigma_\epsilon^2 R} \right). \quad (15)$$

We have $n_t$ exactly equal to $N$ when the current value of $\langle v_t^2 \rangle$ satisfies

$$\langle v_t^2 \rangle < \gamma_c \equiv \frac{\mu \sigma_\epsilon^2 R}{2R(N-1) - \mu(R^2(N-2) - S)} \qquad (n_t = N). \quad (16)$$

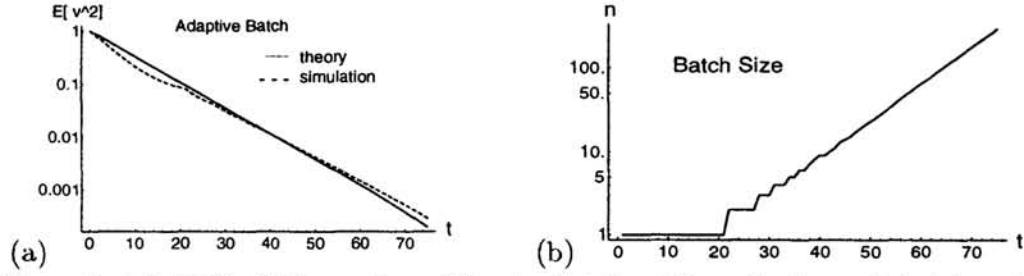

Figure 2: 1-D LMS: a) Comparison of the simulated and theoretically predicted (equation (18)) squared weight error as a function of $t$ with $N = 1000$, $R = 1$, $\sigma_\epsilon = 1$, $\mu = .1$, and 10000 networks. b) Corresponding batch sizes used in the simulations.

Thus, after $\langle v_t^2 \rangle$ has decreased to $\gamma_c$, training will proceed as pure batch. When $\langle v_t^2 \rangle > \gamma_c$, we have $n_t < N$ and we can put (15) into (10) to obtain

$$\langle v_{t+1}^2 \rangle = \left(1 - \mu R + \mu^2 \frac{(NR^2 - S)}{2(N-1)}\right)\langle v_t^2 \rangle - \frac{\mu^2 \sigma_\epsilon^2 R}{2(N-1)}. \tag{17}$$

Solving (17) we get

$$\langle v_t^2 \rangle = \alpha_1^{t-t_0}\langle v_0^2 \rangle + \frac{1 - \alpha_1^{t-t_0}}{1 - \alpha_1}\beta_1 \tag{18}$$

where $\alpha_1$, and $\beta_1$ are constants

$$\alpha_1 = 1 - \mu R + \mu^2 \frac{(S - NR^2)}{2(N-1)}, \qquad \beta_1 = -\mu^2 \frac{\sigma_\epsilon^2 R}{2(N-1)}. \tag{19}$$

Figure 2a compares equation (18) with 1-D LMS simulations. The adaptive batch size was chosen by rounding (15) to the nearest integer. Early in training, the predicted $n_t$ is always smaller than 1 but the simulation always rounds up to 1 (can't have n=0). Figure 2b displays the batch sizes that were used in the simulations. A logarithmic scale is used to show that the batch size increases exponentially in $t$. We next examine the batch size as a function of $\tau$.

### 3.1 Convergence Rate per Input Presentation

When we use (15) to choose the batch size, the number of input presentations will vary at each iteration. Thus, $\tau$ is not simply a multiple of $t$. Instead, we have

$$\tau(t) = \tau_0 + \sum_{i=t_0}^{t} n_i \tag{20}$$

where $\tau_0$ is the number of inputs that have been presented by $t_0$. This can be evaluated when $N$ is very large. In this case, equations (18) and (15) reduce to

$$\langle v_t^2 \rangle = \langle v_0^2 \rangle \alpha_3^{t-t_0} \quad \text{where} \quad \alpha_3 \equiv 1 - \mu R + \frac{1}{2}\mu^2 R^2 \tag{21}$$

$$n_t = \frac{2\mu((S - R^2)\langle v_t^2 \rangle + \sigma_\epsilon^2 R)}{(2R - \mu R^2)\langle v_t^2 \rangle} = \frac{2\mu(S - R^2)}{2R - \mu R^2} + \frac{2\mu\sigma_\epsilon^2}{(2 - \mu R)\langle v_0^2 \rangle \alpha_3^{t-t_0}}. \tag{22}$$

Putting (22) into (20), and summing gives gives

$$\Delta\tau(t) = \frac{2\mu(S - R^2)}{(2 - \mu R)R}\Delta t + \frac{2\mu\sigma_\epsilon^2}{(2 - \mu R)\langle v_0^2 \rangle}\frac{\alpha_3^{-\Delta t} - \alpha_3}{1 - \alpha_3} \tag{23}$$

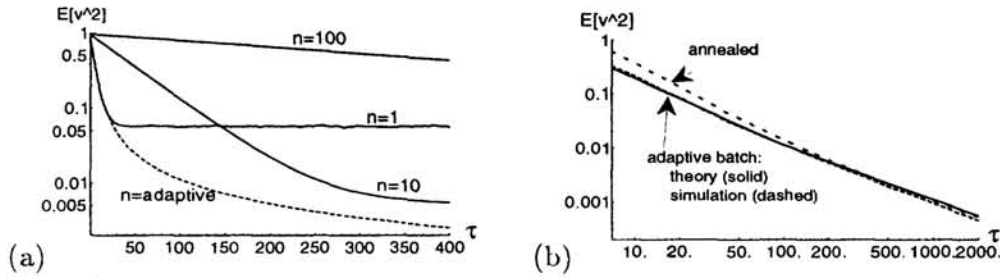

Figure 3: 1-D LMS: a) Simulations of the squared weight error as a function of $\tau$, the number of input presentations for $N = 1000$, $R = 1$, $\sigma_\epsilon = 1$, $\mu = .1$, and 10000 networks. Batch sizes are $n = 1$, 10, 100, and $n_t$ (see (15)). b) Comparison of simulation (dashed) and theory (see (24)) using adaptive batch size. Simulation (long dash) using an annealed learning rate with $n = 1$ and $\mu = R^{-1}$ is also shown.

where $\Delta t \equiv t - t_0$ and $\Delta \tau \equiv \tau - \tau_0$. Assuming that $|\alpha_3| < 1$, the term with $\alpha_3^{-\Delta t}$ will dominate at late times. Dropping the other terms and solving for $\alpha_3^t$ gives

$$\langle v_t^2 \rangle = \langle v_t^2 \rangle \alpha_3^{\Delta t} \approx \frac{4\,\sigma_\epsilon^2}{(2 - \mu R)^2\,R\,(\tau - \tau_0)}. \tag{24}$$

Thus, when using an adaptive batch size, $\langle v^2 \rangle$ converges at late times as $\frac{1}{\tau}$. Figure 3a compares simulations of $\langle v^2 \rangle$ with adaptive and constant batch sizes. As can be seen, the adaptive $n$ curve follows the $n = 1$ curve until just before the $n = 1$ curve starts to flatten out. Figure 3b compares (24) with the simulation. Curves are plotted on a log-log plot to illustrate the $1/\tau$ relationship at late times (straight line with slope of -1).

## 4 Learning Rate Annealing vs Increasing Batch Size

With online learning ($n = 1$), we can reduce the noise at late times by annealing the learning rate using a $\mu/t$ schedule. During this phase, $\langle v^2 \rangle$ decreases at a rate of $1/\tau$ if $\mu > R^{-1}/2$ [LO93] and slower otherwise. In this paper, we have presented an alternative method for reducing the noise by increasing the batch size exponentially in $t$. Here, $\langle v^2 \rangle$ also decreases at rate of $1/\tau$ so that, from this perspective, an adaptive batch size is equivalent to annealing the learning rate. This is confirmed in Figure 3b which compares using an adaptive batch size with annealing.

An advantage of the adaptive batch size comes when $n$ reaches $N$. At this point $n$ remains constant so that $\langle v^2 \rangle$ decreases exponentially in $\tau$. However, with annealing, the convergence rate of $\langle v^2 \rangle$ always remains proportional to $1/\tau$. A disadvantage, though, occurs in multiple dimensions with nonspherical $R$ where the best choice of $n_t$ would likely be different along different directions in the weight space. Though it is possible to have a different learning rate along different directions, it is not possible to have different batch sizes.

## 5 Appendix A

In this appendix we use simple counting arguments to derive the two results in equations (5) and (6). We first note that there are $M \equiv \begin{pmatrix} N \\ n \end{pmatrix}$ ways of choosing $n$ examples out of a total of $N$ examples. Thus, (5) can be rewritten as

$$\langle g_{B_n,t} \rangle_B = \frac{1}{M} \sum_{i=1}^{M} g_{B_n^{(i)},t} = \frac{1}{M} \sum_{i=1}^{M} \frac{1}{n} \sum_{z_j \in B_n^{(i)}} g_{j,t}. \tag{25}$$

where $B_n^{(i)}$ is the $i^{th}$ batch $(i = 1, \ldots, M)$, and $g_{j,t} \equiv \partial \mathcal{E}(z_j)/\partial v_t$ for $j = 1, \ldots, N$. If we were to expand (25) we would find that there are exactly $nM$ terms. From symmetry and since there are only $N$ unique $g_{j,t}$, we conclude that each $g_{j,t}$ occurs exactly $\frac{nM}{N}$ times. The above expression can then be written as

$$\langle g_{B_n,t} \rangle_B = \frac{1}{nM} \frac{nM}{N} \sum_{j=1}^{N} g_{j,t} = \frac{1}{N} \sum_{j=1}^{N} g_{j,t} = \langle g_{i,t} \rangle_N. \tag{26}$$

Thus, we have equation (5). The second equation (6) is

$$\langle g_{B_n,t}^2 \rangle_B = \frac{1}{M} \sum_{i=1}^{M} g_{B_n,t}^T g_{B_n,t} = \frac{1}{M} \sum_{i=1}^{M} \left( \frac{1}{n} \sum_{z_j \in B_n^{(i)}} g_{j,t}^T \right) \left( \frac{1}{n} \sum_{z_k \in B_n^{(i)}} g_{k,t} \right)$$

$$= \frac{1}{n^2 M} \sum_{i=1}^{M} \sum_{z_j, z_k \in B_n^{(i)}} g_{j,t}^T g_{k,t} = \frac{1}{n^2 M} \sum_{i=1}^{M} \left( \sum_{z_j \in B_n^{(i)}} g_{j,t}^2 + \sum_{\substack{z_j, z_k \in B_n^{(i)} \\ j \neq k}} g_{j,t}^T g_{k,t} \right). \tag{27}$$

By the same argument to derive (5), the first term on the right $(1/n)\langle g_{i,t}^2 \rangle_N$. In the second term, there are a total $n(n-1)M$ terms in the sum, of which only $N(N-1)$ are unique. Thus, a given $g_{j,t}^T g_{k,t}$ occurs exactly $n(n-1)M/(N(N-1))$ times so that

$$\frac{1}{n^2 M} \sum_{i=1}^{M} \sum_{z_j, z_k \in B_n^{(i)}, j \neq k} g_{j,t} \, g_{k,t} = \frac{1}{n^2 M} \frac{n(n-1)M}{N(N-1)} \sum_{j,k=1, j \neq k}^{N} g_{j,t} \, g_{k,t}$$

$$= \frac{N(n-1)}{n(N-1)} \left( \frac{1}{N^2} \cdot \left( \sum_{j,k=1, j \neq k}^{N} g_{j,t} \, g_{k,t} + \sum_{j=1}^{N} g_{j,t}^2 \right) - \frac{1}{N} \left( \frac{1}{N} \sum_{j=1}^{N} g_{j,t}^2 \right) \right)$$

$$= \frac{N(n-1)}{n(N-1)} \langle g_{i,t} \rangle_N^2 - \frac{(n-1)}{n(N-1)} \langle g_{i,t}^2 \rangle_N. \tag{28}$$

Putting the simplified first term together and (28) both into (27) we obtain our second result in equation (6).

## Footnotes

[1] For real zero mean gaussian inputs, we can apply the gaussian moment factoring theorem [Hay91] which states that $\langle x_i x_j x_k x_l \rangle_N = R_{ij} R_{kl} + R_{ik} R_{jl} + R_{il} R_{jk}$ where the subscripts on $x$ denote components of $x$. From this, we find that $S = (\text{Trace } R)R + 2R^2$.

## References

[Gol87] Larry Goldstein. Mean square optimality in the continuous time Robbins Monro procedure. Technical Report DRB-306, Dept. of Mathematics, University of Southern California, LA, 1987.

[Hay91] Simon Haykin. *Adaptive Filter Theory*. Prentice Hall, New Jersey, 1991.

[LO93] Todd K. Leen and Genevieve B. Orr. Momentum and optimal stochastic search. In *Advances in Neural Information Processing Systems, vol. 6*, 1993. to appear.

[Møl93] Martin Møller. Supervised learning on large redundant training sets. *International Journal of Neural Systems*, 4(1):15–25, 1993.

[Orr95] Genevieve B. Orr. *Dynamics and Algorithms for Stochastic learning*. PhD thesis, Oregon Graduate Institute, 1995.

[PW93] Mark Plutowski and Halbert White. Selecting concise training sets from clean data. *IEEE Transactions on Neural Networks*, 4:305–318, 1993.